# Representing Part-Whole Relationships in Recurrent Neural Networks

**Viren Jain[2], Valentin Zhigulin[1,2], and H. Sebastian Seung[1,2]**
[1]Howard Hughes Medical Institute and
[2]Brain & Cog. Sci. Dept., MIT
`viren@mit.edu, valentin@mit.edu, seung@mit.edu`

## Abstract

There is little consensus about the computational function of top-down synaptic connections in the visual system. Here we explore the hypothesis that top-down connections, like bottom-up connections, reflect part-whole relationships. We analyze a recurrent network with bidirectional synaptic interactions between a layer of neurons representing parts and a layer of neurons representing wholes. Within each layer, there is lateral inhibition. When the network detects a whole, it can rigorously enforce part-whole relationships by ignoring parts that do not belong. The network can complete the whole by filling in missing parts. The network can refuse to recognize a whole, if the activated parts do not conform to a stored part-whole relationship. Parameter regimes in which these behaviors happen are identified using the theory of permitted and forbidden sets [3, 4]. The network behaviors are illustrated by recreating Rumelhart and McClelland's "interactive activation" model [7].

In neural network models of visual object recognition [2, 6, 8], patterns of synaptic connectivity often reflect part-whole relationships between the features that are represented by neurons. For example, the connections of Figure 1 reflect the fact that feature B both contains simpler features A1, A2, and A3, and is contained in more complex features C1, C2, and C3. Such connectivity allows neurons to follow the rule that existence of the part is evidence for existence of the whole. By combining synaptic input from multiple sources of evidence for a feature, a neuron can "decide" whether that feature is present. [1]

The synapses shown in Figure 1 are purely bottom-up, directed from simple to complex features. However, there are also top-down connections in the visual system, and there is little consensus about their function. One possibility is that top-down connections also reflect part-whole relationships. They allow feature detectors to make decisions using the rule that existence of the whole is evidence for existence of its parts.

In this paper, we analyze the dynamics of a recurrent network in which part-whole relationships are stored as bidirectional synaptic interactions, rather than the unidirectional interactions of Figure 1. The network has a number of interesting computational capabilities. When the network detects a whole, it can rigorously enforce part-whole relationships

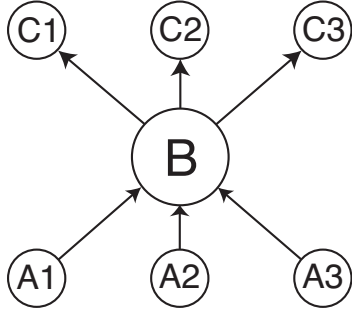

Figure 1: The synaptic connections (arrows) of neuron B represent part-whole relationships. Feature B both contains simpler features and is contained in more complex features. The synaptic interactions are drawn one-way, as in most models of visual object recognition. Existence of the part is regarded as evidence for existence of the whole. This paper makes the interactions bidirectional, allowing the existence of the whole to be evidence for the existence of its parts.

by ignoring parts that do not belong. The network can complete the whole by filling in missing parts. The network can refuse to recognize a whole, if the activated parts do not conform to a stored part-whole relationship. Parameter regimes in which these behaviors happen are identified using the recently developed theory of permitted and forbidden sets [3, 4].

Our model is closely related to the interactive activation model of word recognition, which was proposed by McClelland and Rumelhart to explain the word superiority effect studied by visual psychologists [7]. Here our concern is not to model a psychological effect, but to characterize mathematically how computations involving part-whole relationships can be carried out by a recurrent network.

# 1 Network model

Suppose that we are given a set of part-whole relationships specified by

$$\xi_i^a = \begin{cases} 1, & \text{if part } i \text{ is contained in whole } a \\ 0, & \text{otherwise} \end{cases}$$

We assume that every whole contains at least one part, and every part is contained in at least one whole.

The stimulus drives a layer of neurons that detect parts. These neurons also interact with a layer of neurons that detect wholes. We will refer to part-detectors as "P-neurons" and whole-detectors as "W-neurons."

The part-whole relationships are directly stored in the synaptic connections between P and W neurons. If $\xi_i^a = 1$, the $i$th neuron in the P layer and the $a$th neuron in the W layer have an excitatory interaction of strength $\gamma$. If $\xi_i^a = 0$, the neurons have an inhibitory interaction of strength $\sigma$. Furthermore, the P-neurons inhibit each other with strength $\beta$, and the W-neurons inhibit each other with strength $\alpha$. All of these interactions are symmetric, and all activation functions are the rectification nonlinearity $[z]^+ = \max\{z, 0\}$.

Then the dynamics of the network takes the form

$$\dot{W}_a + W_a = \left[ \gamma \sum_i P_i \xi_i^a - \sigma \sum_i (1 - \xi_i^a) P_i - \alpha \sum_{b \neq a} W_b \right]^+, \tag{1}$$

$$\dot{P}_i + P_i = \left[ \gamma \sum_a W_a \xi_i^a - \sigma \sum_a (1 - \xi_i^a) W_a - \beta \sum_{j \neq i} P_j + B_i \right]^+. \tag{2}$$

where $B_i$ is the input to the P layer from the stimulus. Figure 2 shows an example of a network with two wholes. Each whole contains two parts. One of the parts is contained in both wholes.

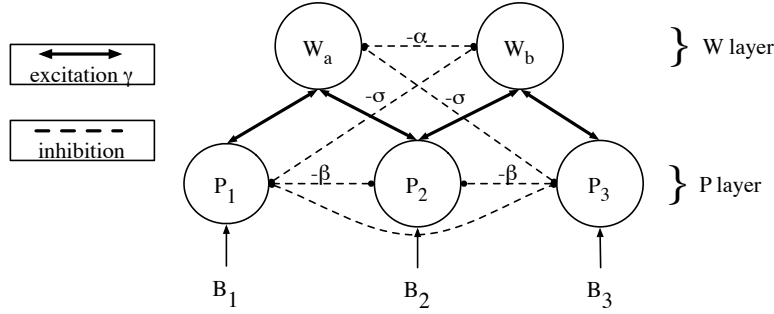

Figure 2: Model in example configuration: $\xi = \{(1, 1, 0), (0, 1, 1)\}$.

When a stimulus is presented, it activates some of the P-neurons, which activate some of the W-neurons. The network eventually converges to a stable steady state. We will assume that $\alpha > 1$. In the Appendix, we prove that this leads to unconditional winner-take-all behavior in the W layer. In other words, no more than one W-neuron can be active at a stable steady state.

If a single W-neuron is active, then a whole has been detected. Potentially there are also many P-neurons active, indicating detection of parts. This representation may have different properties, depending on the choice of parameters $\beta$, $\gamma$, and $\sigma$. As discussed below, these include rigorous enforcement of part-whole relationships, completion of wholes by "filling in" missing parts, and non-recognition of parts that do not conform to a whole.

## 2 Enforcement of part-whole relationships

Suppose that a single W-neuron is active at a stable steady state, so that a whole has been detected. Part-whole relationships are said to be enforced if the network always ignores parts that are not contained in the detected whole, despite potentially strong bottom-up evidence for them. It can be shown that enforcement follows from the inequality

$$\sigma^2 + \beta^2 + \gamma^2 + 2\sigma\beta\gamma > 1. \tag{3}$$

which guarantees that neuron $i$ in the P layer is inactive, if neuron $a$ in the W layer is active and $\xi_i^a = 0$. When part-whole relations are enforced, prior knowledge about legal combinations of parts strictly constrains what may be perceived. This result is proven in the Appendix, and only an intuitive explanation is given here.

Enforcement is easiest to understand when there is interlayer inhibition ($\sigma > 0$). In this case, the active W-neuron directly inhibits the forbidden P-neurons. The case of $\sigma = 0$ is more subtle. Then enforcement is mediated by lateral inhibition in the P layer. Excitatory feedback from the W-neuron has the effect of counteracting the lateral inhibition between the P-neurons that belong to the whole. As a result, these P-neurons become strongly activated enough to inhibit the rest of the P layer.

## 3 Completion of wholes by filling in missing parts

If a W-neuron is active, it excites the P-neurons that belong to the whole. As a result, even if one of these P-neurons receives no bottom-up input ($B_i = 0$), it is still active. We call

this phenomenon "completion," and it is guaranteed to happen when

$$\gamma > \sqrt{\beta} \qquad (4)$$

The network may thus "imagine" parts that are consistent with the recognized whole, but are not actually present in the stimulus. As with enforcement, this condition depends on top-down connections.

In the special case $\gamma = \sqrt{\beta}$, the interlayer excitation between a W-neuron and its P-neurons exactly cancels out the lateral inhibition between the P-neurons at a steady state. So the recurrent connections effectively vanish, letting the activity of the P-neurons be determined by their feedforward inputs. When the interlayer excitation is stronger than this, the inequality (4) holds, and completion occurs.

## 4   Non-recognition of a whole

If there is no interlayer inhibition ($\sigma = 0$), then a single W-neuron is always active, assuming that there is some activity in the P layer. To see this, suppose for the sake of contradiction that all the W-neurons are inactive. Then they receive no inhibition to counteract the excitation from the P layer. This means some of them must be active, which contradicts our assumption. This means that the network always recognizes a whole, even if the stimulus is very different from any part-whole combination that is stored in the network.

However, if interlayer inhibition is sufficiently strong (large $\sigma$), the network may refuse to recognize a whole. Neurons in the P layer are activated, but there is no activity in the W layer. Formal conditions on $\sigma$ can be derived, but are not given here because of space limitations.

In case of non-recognition, constraints on the P-layer are not enforced. It is possible for the network to detect a configuration of parts that is not consistent with any stored whole.

## 5   Example: Interactive Activation model

To illustrate the computational capabilities of our network, we use it to recreate the interactive activation (IA) model of McClelland and Rumelhart. Figure 3 shows numerical simulations of a network containing three layers of neurons representing strokes, letters, and words, respectively. There are 16 possible strokes in each of four letter positions. For each stroke, there are two neurons, one signaling the presence of the stroke and the other signaling its absence. Letter neurons represent each letter of the alphabet in each of four positions. Word neurons represent each of 1200 common four letter words.

The letter and word layers correspond to the P and W layers that were introduced previously. There are bidirectional interactions between the letter and word layers, and lateral inhibition within the layers. The letter neurons also receive input from the stroke neurons, but this interaction is unidirectional.

Our network differs in two ways from the original IA model. First, all interactions involving letter and word neurons are symmetric. In the original model, the interactions between the letter and word layers were asymmetric. In particular, inhibitory connections only ran from letter neurons to word neurons, and not vice versa. Second, the only nonlinearity in our model is rectification. These two aspects allow us to apply the full machinery of the theory of permitted and forbidden sets.

Figure 3 shows the result of presenting the stimulus "MO M" for four different settings of parameters. In each of the four cases, the word layer of the network converges to the same result, detecting the word "MOON", which is the closest stored word to the stimulus. However, the activity in the letter layer is different in the four cases.

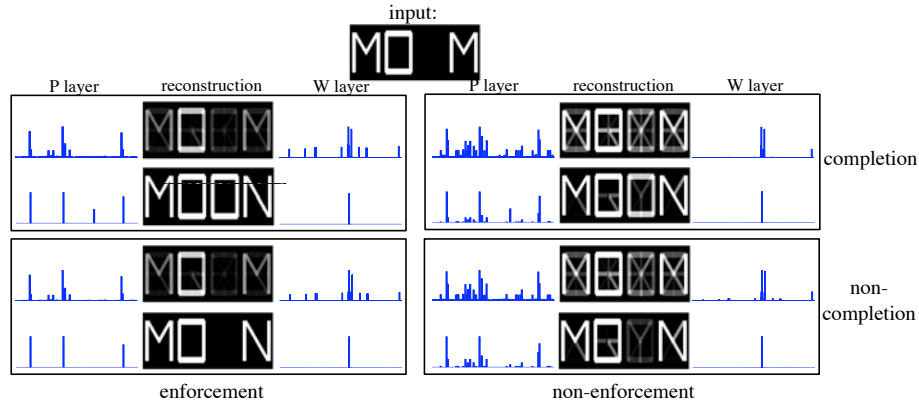

Figure 3: Simulation of 4 different parameter regimes in a letter-word recognition network. Within each panel, the middle column presents a feature-layer reconstruction based on the letter activity shown in the left column. W layer activity is shown in the right column. The top row shows the network state after 10 iterations of the dynamics. The bottom row shows the steady state.

In the left column, the parameters obey the inequality (3), so that part-whole relationships are enforced. The activity of the letter layer is visualized by activating the strokes corresponding to each active letter neuron. The activated letters are part of the word "MOON". In the top left, the inequality (4) is satisfied, so that the missing "O" in the stimulus is filled in. In the bottom left, completion does not occur.

In the simulations of the right column, parameters are such that part-whole relationships are not enforced. Consequently, the word layer is much more active. Bottom-up input provides evidence for several other letters, which is not suppressed. In the top right, the inequality (4) is satisfied, so that the missing "O" in the stimulus is filled in. In the bottom right, the "O" neuron is not activated in the third position, so there is no completion. However, some letter neurons for the third position are activated, due to the input from neurons that indicate the absence of strokes.

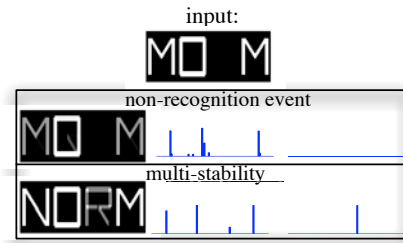

Figure 4: Simulation of a non-recognition event and example of multistability.

Figure 4 shows simulations for large $\sigma$, deep in the enforcement regime where non-recognition is a possibility. From one initial condition, the network converges to a state in which no W neurons are active, a non-recognition. From another initial condition, the network detects the word "NORM". Deep in the enforcement regime, the top-down feedback can be so strong that the network has multiple stable states, many of which bear little resemblance to the stimulus at all. This is a problematic aspect of this network. It can be prevented by setting parameters at the edge of the enforcement regime.

## 6 Discussion

We have analyzed a recurrent network that performs computations involving part-whole relationships. The network can fill in missing parts and suppress parts that do not belong.

These two computations are distinct and can be dissociated from each other, as shown in Figure 3.

While these two computations can also be performed by associative memory models, they are not typically dissociable in these models. For example, in the Hopfield model pattern completion and noise suppression are both the result of recall of one of a finite number of stereotyped activity patterns.

We believe that our model is more appropriate for perceptual systems, because its behavior is piecewise linear, due its reliance on rectification nonlinearity. Therefore, analog aspects of computation are able to coexist with the part-whole relationships. Furthermore, in our model the stimulus is encoded in maintained synaptic input to the network, rather than as an initial condition of the dynamics.

# A   Appendix: Permitted and forbidden sets

Our mathematical results depend on the theory of permitted and forbidden sets [3, 4], which is summarized briefly here. The theory is applicable to neural networks with rectification nonlinearity, of the form $\dot{x}_i + x_i = [b_i + \sum_j W_{ij}x_j]^+$. Neuron $i$ is said to be active when $x_i > 0$. For a network of $N$ neurons, there are $2^N$ possible sets of active neurons. For each active set, consider the submatrix of $W_{ij}$ corresponding to the synapses between active neurons. If all eigenvalues of this submatrix have real parts less than or equal to unity, then the active set is said to be *permitted*. Otherwise the active set is said to be *forbidden*. A set is permitted if and only if there exists an input vector $b$ such that those neurons are active at a stable steady state. Permitted sets can be regarded as memories stored in the synaptic connections $W_{ij}$. If $W_{ij}$ is a symmetric matrix, the *nesting property* holds: every subset of a permitted set is permitted, and every superset of a forbidden set is forbidden.

The present model can be seen as a general method for storing permitted sets in a recurrent network. This method introduces a neuron for each permitted set, relying on a unary or "grandmother cell" representation. In contrast, Xie et al.[9] used lateral inhibition in a single layer of neurons to store permitted sets. By introducing extra neurons, the present model achieves superior storage capacity, much as unary models of associative memory [1] surpass distributed models [5].

## A.1   Unconditional winner-take-all in the W layer

The synapses between two W-neurons have strengths

$$\begin{pmatrix} 0 & -\alpha \\ -\alpha & 0 \end{pmatrix}$$

The eigenvalues of this matrix are $\pm\alpha$. Therefore two W-neurons constitute a forbidden set if $\alpha > 1$. By the nesting property, it follows more than two W-neurons is also a forbidden set, and that the W layer has the unconditional winner-take-all property.

## A.2   Part-whole combinations as permitted sets

**Theorem 1.** *Suppose that $\beta < 1$. If $\gamma^2 < \beta + (1-\beta)/k$ then any combination of $k \geq 1$ parts consistent with a whole corresponds to a permitted set.*

*Proof.* Consider $k$ parts belonging to a whole. They are represented by one W-neuron and $k$ P-neurons, with synaptic connections given by the $(k+1) \times (k+1)$ matrix

$$M = \begin{pmatrix} -\beta(\mathbf{1}\mathbf{1}^T - I) & \gamma\mathbf{1} \\ \gamma\mathbf{1}^T & 0 \end{pmatrix}, \tag{5}$$

where $\mathbf{1}$ is the $k$-dimensional vector whose elements are all equal to one. Two eigenvectors of $M$ are of the form $(\mathbf{1}^T c)$, and have the same eigenvalues as the $2 \times 2$ matrix

$$\begin{pmatrix} -\beta(k-1) & \gamma \\ \gamma k & 0 \end{pmatrix}$$

This matrix has eigenvalues less than one when $\gamma^2 < \beta + (1-\beta)/k$ and $\beta(k-1) + 2 > 0$. The other $k-1$ eigenvectors are of the form $(d^T, 0)$, where $d^T \mathbf{1} = 0$. These have eigenvalues $\beta$. Therefore all eigenvalues of $W$ are less than one if the condition of the theorem is satisfied. $\qquad \square$

### A.3 Constraints on combining parts

Here, we derive conditions under which the network can enforce the constraint that steady state activity be confined to parts that constitute a whole.

**Theorem 2.** *Suppose that $\beta > 0$ and $\sigma^2 + \beta^2 + \gamma^2 + 2\sigma\beta\gamma > 1$ If a W-neuron is active, then only P-neurons corresponding to parts contained in the relevant whole can be active at a stable steady state.*

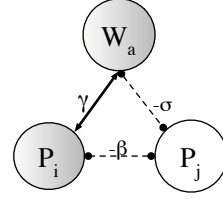

Figure 5: A set of one W-neuron and two P-neurons is forbidden if one part belongs to the whole and the other does not.

*Proof.* Consider P-neurons $P_i$, $P_j$, and W-neuron $W_a$. Suppose that $\xi_i^a = 1$ but $\xi_j^a = 0$. As shown in Figure 5, the matrix of connections is given by:

$$W = \begin{pmatrix} 0 & -\beta & \gamma \\ -\beta & 0 & -\sigma \\ \gamma & -\sigma & 0 \end{pmatrix} \qquad (6)$$

This set is permitted if all eigenvalues of $W - I$ have negative real parts. The characteristic equation of $I - W$ is $\lambda^3 + b_1\lambda^2 + b_2\lambda + b_3 = 0$, where $b_1 = 3$, $b_2 = 3 - \sigma^2 - \beta^2 - \gamma^2$ and $b_3 = 1 - 2\sigma\beta\gamma - \sigma^2 - \beta^2 - \gamma^2$. According to the Routh-Hurwitz theorem, all the eigenvalues have negative real parts if and only if $b_1 > 0$, $b_3 > 0$ and $b_1 b_2 > b_3$. Clearly, the first condition is always satisfied. The second condition is more restrictive than the third. It is satisfied only when $\sigma^2 + \beta^2 + \gamma^2 + 2\sigma\beta\gamma < 1$. Hence, one of the eigenvalues has a positive real part when this condition is broken, i.e., when $\sigma^2 + \beta^2 + \gamma^2 + 2\sigma\beta\gamma > 1$. By the nesting property, any larger set of P-neurons inconsistent with the W-neuron is also forbidden. $\quad \square$

### A.4 Completion of wholes

**Theorem 3.** *If $\gamma > \sqrt{\beta}$ and a single W-neuron $a$ is active at a steady state, then $P_i > 0$ for all $i$ such that $\xi_i^a = 1$.*

*Proof.* Suppose that the detected whole has $k$ parts. At the steady state

$$P_i = \frac{\xi_i^a}{1-\beta} \left[ B_i - (\beta - \gamma^2) P_{tot} \right]^+$$

where

$$P_{tot} = \sum_i P_i = \frac{1}{1 - \beta + (\beta - \gamma^2)k} \sum_{i=1}^{k} B_i \xi_i^a \qquad (7)$$

$\square$

### A.5 Preventing runaway

If feedback loops cause the network activity to diverge, then the preceding analyses are not relevant. Here we give a sufficient condition guaranteeing that runaway instability does not happen. It is not a necessary condition. Interestingly, the condition implies the condition of Theorem 1.

**Theorem 4.** *Suppose that $P$ and $W$ obey the dynamics of Eqs. (1) and (2), and define the objective function*

$$
\begin{aligned}
E \quad = \quad & \frac{1-\alpha}{2}\sum_a W_a^2 + \frac{\alpha}{2}\left(\sum_a W_a\right)^2 + \frac{1-\beta}{2}\sum_i P_i^2 + \frac{\beta}{2}\left(\sum_i P_i\right)^2 \\
& - \sum_i B_i P_i - \gamma \sum_{ia} P_i W_a \xi_i^a + \sigma \sum_{ia}(1-\xi_i^a)P_i W_a.
\end{aligned}
\tag{8}
$$

*Then $E$ is a Lyapunov like function that, given $\beta > \gamma^2 - \frac{1-\gamma^2}{N-1}$, ensures convergence of the dynamics to a stable steady state.*

*Proof.* (sketch) Differentiation of $E$ with respect to time shows that that $E$ is nonincreasing in the nonnegative orthant and constant only at steady states of the network dynamics. We must also show that $E$ is radially unbounded, which is true if the quadratic part of $E$ is copositive definite. Note that the last term of $E$ is lower-bounded by zero and the previous term is upper bounded by $\gamma \sum_{ia} P_i W_a$. We assume $\alpha > 1$. Thus, we can use Cauchy's inequality, $\sum_i P_i^2 \geq \left(\sum_i P_i\right)^2/N$, and the fact that $\sum_a W_a^2 \leq (\sum_a W_a)^2$ for $W_a \geq 0$, to derive

$$
E \geq \frac{1}{2}\left((\sum_a W_a)^2 + \frac{1-\beta+\beta N}{N}(\sum_i P_i)^2 - 2\gamma(\sum_a W_a \sum_i P_i)\right) - \sum_i B_i P_i. \tag{9}
$$

If $\beta > \gamma^2 - \frac{1-\gamma^2}{N-1}$, the quadratic form in the inequality is positive definite and $E$ is radially unbounded. $\square$

## Footnotes

[1]Synaptic connectivity may reflect other relationships besides part-whole. For example, invariances can be implemented by connecting detectors of several instances of the same feature to the same target, which is consequently an invariant detector of the feature.

## References

[1] E. B. Baum, J. Moody, and F. Wilczek. Internal representations for associative memory. *Biol. Cybern.*, 59:217–228, 1988.

[2] K. Fukushima. Neocognitron: a self organizing neural network model for a mechanism of pattern recognition unaffected by shift in position. *Biol Cybern*, 36(4):193–202, 1980.

[3] R.H. Hahnloser, R. Sarpeshkar, M.A. Mahowald, R.J. Douglas, and H.S. Seung. Digital selection and analogue amplification coexist in a cortex-inspired silicon circuit. *Nature*, 405(6789):947–51, Jun 22 2000.

[4] R.H. Hahnloser, H.S. Seung, and J.-J. Slotine. Permitted and forbidden sets in symmetric threshold-linear networks. *Neural Computation*, 15:621–638, 2003.

[5] J.J. Hopfield. Neural networks and physical systems with emergent collective computational abilities. *Proc Natl Acad Sci U S A*, 79(8):2554–8, Apr 1982.

[6] Y. LeCun, B. Boser, J. S. Denker, D. Henderson, R. E. Howard, W. Hubbard, and L. D. Jackel. Backpropagation applied to handwritten zip code recognition. *Neural Comput.*, 1:541–551, 1989.

[7] J. L. McClelland and D. E. Rumelhart. An interactive activation model of context effects in letter perception: Part i. an account of basic findings. *Psychological Review*, 88(5):375–407, Sep 1981.

[8] M Riesenhuber and T Poggio. Hierarchical models of object recognition in cortex. *Nat Neurosci*, 2(11):1019–25, Nov 1999.

[9] X. Xie, R.H. Hahnloser, and H. S. Seung. Selectively grouping neurons in recurrent networks of lateral inhibition. *Neural Computation*, 14:2627–2646, 2002.
